# Hidden Common Cause Relations in Relational Learning

**Ricardo Silva**[*]
Gatsby Computational Neuroscience Unit
UCL, London, UK WC1N 3AR
rbas@gatsby.ucl.ac.uk

**Wei Chu**
Center for Computational Learning Systems
Columbia University, New York, NY 10115
chuwei@cs.columbia.edu

**Zoubin Ghahramani**
Department of Engineering
University of Cambridge, UK CB2 1PZ
zoubin@eng.cam.ac.uk

## Abstract

When predicting class labels for objects within a relational database, it is often helpful to consider a model for relationships: this allows for information between class labels to be shared and to improve prediction performance. However, there are different ways by which objects can be related within a relational database. One traditional way corresponds to a Markov network structure: each existing relation is represented by an undirected edge. This encodes that, conditioned on input features, each object label is independent of other object labels given its neighbors in the graph. However, there is no reason why Markov networks should be the only representation of choice for symmetric dependence structures. Here we discuss the case when relationships are postulated to exist due to *hidden common causes*. We discuss how the resulting graphical model differs from Markov networks, and how it describes different types of real-world relational processes. A Bayesian nonparametric classification model is built upon this graphical representation and evaluated with several empirical studies.

## 1 Contribution

Prediction problems, such as classification, can be easier when class labels share a sort of relational dependency that is not accounted by the input features [10]. If the variables to be predicted are attributes of objects in a relational database, such dependencies are often postulated from the relations that exist in the database. This paper proposes and evaluates a new method for building classifiers that uses information concerning the relational structure of the problem.

Consider the following standard example, adapted from [3]. There are different webpages, each one labeled according to some class (e.g., "student page" or "not a student page"). Features such as the word distribution within the body of each page can be used to predict each webpage's class. However, webpages do not exist in isolation: there are links connecting them. Two pages having a common set of links is evidence for similarity between such pages. For instance, if $W_1$ and $W_3$ both link to $W_2$, this is commonly considered to be evidence for $W_1$ and $W_3$ having the same class. One way of expressing this dependency is through the following Markov network [5]:

---

[*]Now at the Statistical Laboratory, University of Cambridge. E-mail: silva@statslab.cam.ac.uk

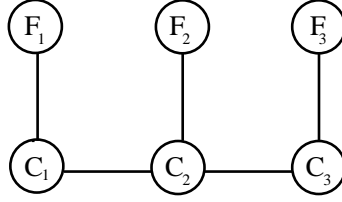

Here $F_i$ are the features of page $W_i$, and $C_i$ is its respective page label. Other edges linking $F$ variables to $C$ variables (e.g., $F_1 - C_2$) can be added without affecting the main arguments presented in this section. The semantics of the graph, for a fixed input feature set $\{F_1, F_2, F_3\}$, are as follows: $C_1$ is marginally *dependent* on $C_3$, but conditionally *independent* given $C_2$. Depending on the domain, this might be either a suitable or unsuitable representation of relations. For instance, in some domains it could be the case that the most sensible model would state that $C_1$ is only informative about $C_3$ once we know what $C_2$ is: that is, $C_1$ and $C_3$ are marginally *independent*, but *dependent* given $C_2$. This can happen if the existence of a relation $(C_i, C_j)$ corresponds to the existence of *hidden common causes* generating this pair of random variables.

Consider the following example, loosely based on a problem described by [12]. We have three objects, Microsoft ($M$), Sony ($S$) and Philips ($P$). The task is a regression task where we want to predict the stock market price of each company given its profitability from last year. The given relationships are that $M$ and $S$ are direct competitors (due to the videogame console market), as well $S$ and $P$ (due to the TV set market).

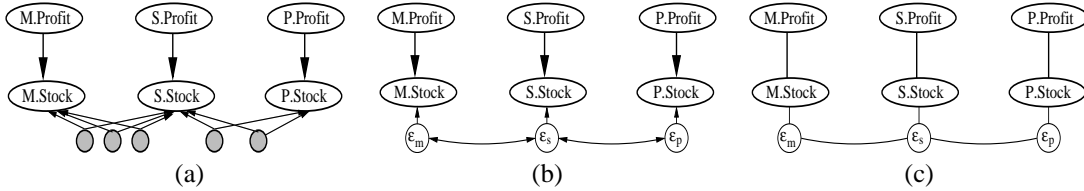

Figure 1: (a) Assumptions that relate Microsoft, Sony and Philips stock prices through hidden common cause mechanisms, depicted as unlabeled gray vertices; (b) A graphical representation for generic hidden common causes relationships by using bi-directed edges; (c) A depiction of the same relationship skeleton by a Markov network model, which has different probabilistic semantics.

It is expected that several market factors that affect stock prices are unaccounted by the predictor variable *Past Year Profit*. For example, a shortage of Microsoft consoles is a hidden common factor for both Microsoft's and Sony's stock. Another hidden common cause would be a high price for Sony's consoles. Assume here that these factors have no effect on Philips' stock value. A depiction of several hidden common causes that correpond to the relations $Competitor(M, S)$ and $Competitor(S, P)$ is given in Figure 1(a) as unlabeled gray vertices.

Consider a linear regression model for this setup. We assume that for each object $O_i \in \{M, S, P\}$, the stock price $O_i.Stock$, centered at the mean, is given by

$$O_i.Stock = \beta \times O_i.Profit + \epsilon_i \tag{1}$$

where each $\epsilon_i$ is a Gaussian random variable.

The fact that there are several hidden common causes between $M$ and $S$ can be modeled by the covariance of $\epsilon_m$ and $\epsilon_s$, $\sigma_{ms}$. That is, unlike in standard directed Gaussian models, $\sigma_{ms}$ is allowed to be non-zero. The same holds for $\sigma_{sp}$. Covariances of error terms of unrelated objects should be zero ($\sigma_{mp} = 0$). This setup is very closely related to the classic *seemingly unrelated regression* model popular in economics [12].

A graphical representation for this type of model is the *directed mixed graph* (DMG) [9, 11], with bi-directed edges representing the relationship of having hidden common causes between a pair of vertices. This is shown in Figure 1(b). Contrast this to the Markov network representation in Figure 1(c). The undirected representation encodes that $\epsilon_m$ and $\epsilon_p$ are marginally dependent, which

does not correspond to our assumptions[1]. Moreover, the model in Figure 1(b) states that once we observe Sony's stock price, Philip's stocks (and profit) should have a non-zero association with Microsoft's profit: this follows from a extension of d-separation to DMGs [9]. This is expected from the assumptions (Philip's stocks should tell us something about Microsoft's once we know Sony's, but not before it), but does not hold in the graphical model in Figure 1(c). While it is tempting to use Markov networks to represent relational models (free of concerns raised by cyclic directed representations), it is clear that there are problems for which they are not a sensible choice.

This is not to say that Markov networks are not the best representation for large classes of relational problems. Conditional random fields [4] are well-motivated Markov network models for sequence learning. The temporal relationship is closed under marginalization: if we do not measure some steps in the sequence, we will still link the corresponding remaining vertices accordingly, as illustrated in Figure 2. Directed mixed graphs are not a good representation for this sequence structure.

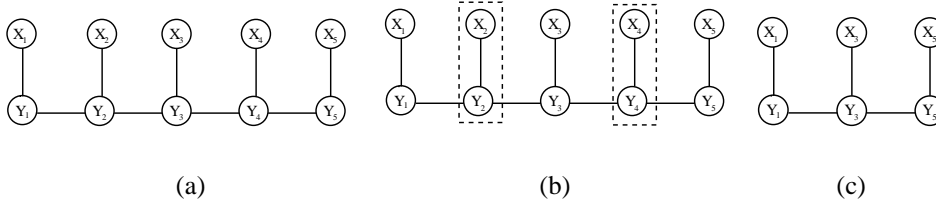

|        (a)        |        (b)        |        (c)        |

Figure 2: (a) A conditional random field (CRF) graph for sequence data; (b) A hypothetical scenario where two of the time slices are not measured, as indicated by dashed boxes; (c) The resulting CRF graph for the remaining variables, which corresponds to the same criteria for construction of (a).

To summarize, the decision between using a Markov network or a DMG reduces to the following modeling issue: if two unlinked object labels $y_i$, $y_j$ are statistically associated when some chain of relationships *exists* between $y_i$ and $y_j$, then the Markov network semantics should apply (as in the case for temporal relationships). However, if the association arises only *given the values* of the other objects in the chain, then this is accounted by the dependence semantics of the directed mixed graph representation. *The DMG representation propagates training data information through other training points. The Markov network representation propagates training data information through test points.* Propagation through training points is relevant in real problems. For instance, in a webpage domain where each webpage has links to pages of several kinds (e.g., [3]), a chain of intermediated points between two classes labels $y_i$ and $y_j$ is likely to be more informative if we know the values of the labels in this chain. The respective Markov network would ignore all training points in this chain besides the endpoints.

In this paper, we introduce a non-parametric classification model for relational data that factorizes according to a directed mixed graph. Sections 2 and 3 describes the model and contrasts it to a closely related approach which bears a strong analogy to the Markov network formulation. Experiments in text classification are described in Section 4.

## 2  Model

Chu et al. [2] describe an approach for Gaussian process classification using relational information, which we review and compare to our proposed model.

**Previous approach: relational Gaussian processes through indicators** − For each point $\mathbf{x}$ in the input space $\mathcal{X}$, there is a corresponding function value $f_{\mathbf{x}}$. Given observed input points $\mathbf{x}_1, \mathbf{x}_2, \ldots, \mathbf{x}_n$, a Gaussian process prior over $\mathbf{f} = [f_1, f_2, \ldots, f_n]^T$ has the shape

$$\mathcal{P}(\mathbf{f}) = \frac{1}{(2\pi)^{n/2}|\Sigma|^{1/2}} \exp\left(-\frac{1}{2}\mathbf{f}^T\Sigma^{-1}\mathbf{f}\right) \qquad (2)$$

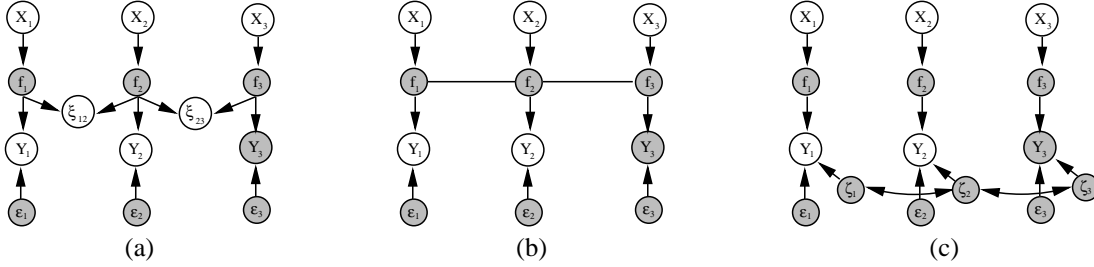

Figure 3: (a) A prediction problem where $y_3$ is unknown and the training set is composed of other two datapoints. Dependencies between $f_1$, $f_2$ and $f_3$ are given by a Gaussian process prior and not represented in the picture. Indicators $\xi_{ij}$ are known and set to 1; (b) The extra associations that arise by conditioning on $\xi = 1$ can be factorized as the Markov network model here depicted, in the spirit of [9]; (c) Our proposed model, which ties the error terms and has origins in known statistical models such as seemingly unrelated regression and structural equation models [11].

where the $ij$th entry of $\Sigma$ is given by a Mercer kernel function $\mathcal{K}(\mathbf{x}_i, \mathbf{x}_j)$ [8].

The idea is to start from a standard Gaussian process prior, and add relational information by conditioning on *relational indicators*. Let $\xi_{ij}$ be an indicator that assumes different values, e.g., 1 or 0. The indicator values are observed for each pair of data points $(\mathbf{x}_i, \mathbf{x}_j)$: they are an encoding of the given relational structure. A model for $P(\xi_{ij} = 1|f_i, f_j)$ is defined. This evidence is incorporated into the Gaussian process by conditioning on all indicators $\xi_{ij}$ that are positive. Essentially, the idea boils down to using $\mathcal{P}(\mathbf{f}|\xi = 1)$ as the prior for a Gaussian process classifier. Figure 3(a) illustrates a problem with datapoints $\{(x_1, y_1), (x_2, y_2), (x_3, y_3)\}$. Gray vertices represent unobserved variables. Each $y_i$ is a binary random variable, with conditional probability given by

$$\mathcal{P}(y_i = 1|f_i) = \Phi(f_i/\sigma) \tag{3}$$

where $\Phi(\cdot)$ is the standard normal cumulative function and $\sigma$ is a hyperparameter. This can be interpreted as the cumulative distribution of $f_i + \epsilon_i$, where $f_i$ is given and $\epsilon_i$ is a normal random variable with zero mean and variance $\sigma^2$.

In the example of Figure 3(a), one has two relations: $(x_1, x_2), (x_2, x_3)$. This information is incorporated by conditioning on the evidence $(\xi_{12} = 1, \xi_{23} = 1)$. Observed points $(x_1, y_1), (x_2, y_2)$ form the training set. The prediction task is to estimate $y_3$. Notice that $\xi_{12}$ is not used to predict $y_3$: the Markov blanket for $f_3$ includes $(f_1, f_2, \xi_{23}, y_3, \epsilon_3)$ and the input features. Essentially, conditioning on $\xi = 1$ corresponds to a pairwise Markov network structure, as depicted in Figure 3(b) [9][2].

**Our approach: mixed graph relational model** − Figure 3(c) illustrates our proposed setup. For reasons that will become clear in the sequel, we parameterize the conditional probability of $y_i$ as

$$\mathcal{P}(y_i = 1|g_i, v_i) = \Phi(g_i/\sqrt{v_i}) \tag{4}$$

where $g_i = f_i + \zeta_i$. As before, Equation (4) can be interpreted as the cumulative distribution of $g_i + \epsilon_i^\star$, with $\epsilon_i^\star$ as a normal random variable with zero mean and variance $v_i = \sigma^2 - \sigma_{\zeta_i}^2$, the last term being the variance of $\zeta_i$. That is, we break the original error term as $\epsilon_i = \zeta_i + \epsilon_i^\star$, where $\epsilon_i^\star$ and $\epsilon_j^\star$ are independent for all $i \neq j$. Random vector $\zeta$ is a multivariate normal with zero mean and covariance matrix $\Sigma_\zeta$. The key aspect in our model is that *the covariance of $\zeta_i$ and $\zeta_j$ is non-zero only if objects $i$ and $j$ are related* (that is, bi-directed edge $y_i \leftrightarrow y_j$ is in the relational graph). Parameterizing $\Sigma_\zeta$ for relational problems is non-trivial and discussed in the next section.

In the example of Figure 3, one noticeable difference of our model 3(c) to a standard Markov network models 3(b) is that now the Markov blanket for $f_3$ includes error terms for all variables (both $\epsilon$ and $\zeta$ terms), following the motivation presented in Section 1.

As before, the prior for $\mathbf{f}$ in our setup is the Gaussian process prior (2). This means that $\mathbf{g}$ has the following Gaussian process prior (implicitly conditioned on $\mathbf{x}$):

$$\mathcal{P}(\mathbf{g}) = \frac{1}{(2\pi)^{n/2}|\mathbf{R}|^{1/2}} \exp\left\{ -\frac{1}{2}\mathbf{g}^\top \mathbf{R}^{-1}\mathbf{g} \right\} \tag{5}$$

where $\mathbf{R} = \mathbf{K} + \Sigma_\zeta$ is the covariance matrix of $\mathbf{g} = \mathbf{f} + \zeta$, with $\mathbf{K}_{ij} = \mathcal{K}(\mathbf{x}_i, \mathbf{x}_j)$.

## 3 Parametrizing a mixed graph model for relational classification

For simplicity, in this paper we will consider only relationships that induce positive associations between labels. Ideally, the parameterization of $\Sigma_\zeta$ has to fulfill two desiderata: (i). it should respect the marginal independence constraints as encoded by the graphical model (i.e., zero covariance for vertices that are not adjacent), and be positive definite; (ii). it has to be parsimonious in order to facilitate hyperparameter selection, both computationally and statistically. Unlike the multivariate analysis problems in [11], the size of our covariance matrix grows with the number of data points.

As shown by [11], exact inference in models with covariance matrices with zero-entry constraints is computationally demanding. We provide two alternative parameterizations that are not as flexible, but which lead to covariance matrices that are simple to compute and easy to implement. We will work under the transductive scenario, where training and all test points are given in advance. The corresponding graph thus contain unobserved and observed label nodes.

### 3.1 Method I

The first method is an automated method to relax some of the independence constraints, while guaranteeing positive-definiteness, and a parameterization that depends on a single scalar $\rho$. This allows for more efficient inference and is done as follows:

1. Let $G_\zeta$ be the corresponding bi-directed subgraph of our original mixed graph, and let $\mathbf{U}^0$ be a matrix with $n \times n$ entries, $n$ being the number of nodes in $G_\zeta$
2. Set $\mathbf{U}^0_{ij}$ to be the number of cliques in $G_\zeta$ where $y_i$ and $y_j$ appear together;
3. Set $\mathbf{U}^0_{ii}$ to be the number of cliques containing $y_i$, plus a small constant $\Delta$;
4. Set $\mathbf{U}$ to be the corresponding correlation matrix obtained by intepreting $\mathbf{U}^0$ as a covariance matrix and rescaling it;

Finally, set $\Sigma_\zeta = \rho\mathbf{U}$, where $\rho \in [0, 1]$ is a given hyperparameter. Matrix $\mathbf{U}$ is always guaranteed to be positive definite: it is equivalent to obtaining the covariance matrix of $\mathbf{y}$ from a linear latent variable model, where there is an independent standard Gaussian latent variable as a common parent to every clique, and every observed node $y_i$ is given by the sum of its parents plus an independent error term of variance $\Delta$. Marginal independencies are respected, since independent random variables will never be in a same clique in $G_\zeta$. In practice, this method cannot be used as is since the number of cliques will in general grow at an exponential rate as a function of $n$. Instead, we first triangulate the graph: in this case, extracting cliques can be done in polynomial time. This is a relaxation of the original goal, since some of the original marginal independence constraints will not be enforced due to the triangulation[3].

### 3.2 Method II

The method suggested in the previous section is appealing under the assumption that vertices that appear in many common cliques are more likely to have more hidden common causes, and hence should have stronger associations. However, sometimes the triangulation introduces bad artifacts, with lots of marginal independence constraints being violated. In this case, this will often result in a poor prediction performance. A cheap alternative approach is not generating cliques, and instead

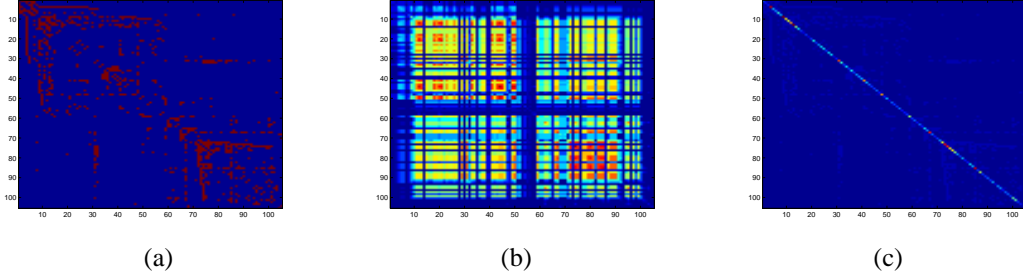

<div align="center">(a)          (b)          (c)</div>

Figure 4: (a) The link matrix for the political books dataset. (b) The relational kernel matrix obtained with the approximated Method I. (c) The kernel matrix obtained with Method II, which tends to produce much weaker associations but does not introduce spurious relations.

getting a marginal covariance matrix from a different latent variable model. In this model, we create an independent standard Gaussian variable for each edge $y_i \leftrightarrow y_j$ instead of each clique. No triangulation will be necessary, and all marginal independence constraints will be respected. This, however, has shortcomings of its own: for all pairs $(y_i, y_j)$ connected by an edge, it will be the case that $\mathbf{U}_{ij}^0 = 1$, while $\mathbf{U}_{ii}^0$ can be as large as $n$. This means that the resulting correlation in $\mathbf{U}_{ij}$ can be close to zero even if $y_i$ and $y_j$ are always in the same cliques. In Section 4, we will choose between Methods I and II according to the marginal likelihood of the model.

### 3.3 Algorithm

Recall that our model is a Gaussian process classifier with error terms $\epsilon_i$ of variance $\sigma$ such that $\epsilon_i = \zeta_i + \epsilon_i^\star$. Without loss of generality, we will assume that $\sigma = 1$. This results in the following parameterization of the full error covariance matrix:

$$\Sigma_\epsilon = (1 - \rho)\mathbf{I} + \rho\mathbf{U} \qquad (6)$$

where $\mathbf{I}$ is an $n \times n$ identity matrix. Matrix $(1 - \rho)\mathbf{I}$ corresponds to the covariance matrix $\Sigma_{\epsilon^\star}$.

The usefulness of separating $\epsilon$ as $\epsilon^\star$ and $\zeta$ becomes evident when we use an expectation-propagation (EP) algorithm [7] to perform inference in our relational classifier. Instead of approximating the posterior of $\mathbf{f}$, we approximate the posterior density $\mathcal{P}(\mathbf{g}|\mathcal{D})$, $\mathcal{D} = \{(\mathbf{x}_1, y_1), \ldots, (\mathbf{x}_n, y_n)\}$ being the given training data. The approximate posterior has the form $\mathcal{Q}(\mathbf{g}) \propto \mathcal{P}(\mathbf{g}) \prod_i \tilde{t}_i(g_i)$ where $\mathcal{P}(\mathbf{g})$ is the Gaussian process prior with kernel matrix $\mathbf{R} = \mathbf{K} + \Sigma_\zeta$ as defined in the previous section. Since the covariance matrix $\Sigma_{\epsilon^\star}$ is diagonal, the true likelihood of $\mathbf{y}$ given $\mathbf{g}$ factorizes over each datapoint: $\mathcal{P}(\mathbf{y}|\mathbf{g}) = \prod_{i=1}^{n} \mathcal{P}(y_i|g_i)$, and standard EP algorithms for Gaussian process classification can be used [8] (with the variance given by $\Sigma_{\epsilon^\star}$ instead of $\Sigma_\epsilon$, and kernel matrix $\mathbf{R}$ instead of $\mathbf{K}$).

The final algorithm defines a whole new class of relational models, depends on a single hyperparameter $\rho$ which can be optimized by grid search in $[0, 1]$, and requires virtually no modification of code written for EP-based Gaussian process classifiers[4].

## 4 Results

We now compare three different methods in relational classification tasks. We will compare a standard Gaussian process classifier (GPC), the relational Gaussian process (RGP) of [2] and our method, the mixed graph Gaussian process (XGP). A linear kernel $\mathcal{K}(\mathbf{x}, \mathbf{z}) = \mathbf{x} \cdot \mathbf{z}$ is used, as described by [2]. We set $\Delta = 10^{-4}$ and the hyperparameter $\rho$ is found by a grid search in the space $\{0.1, 0.2, 0.3, \ldots, 1.0\}$ maximizing the approximate EP marginal likelihood[5].

Table 1: The averaged AUC scores of citation prediction on test cases of the Cora database are recorded along with standard deviation over 100 trials. "$n$" denotes the number of papers in one class. "Citations" denotes the citation count within the two paper classes.

| Group | $n$ | Citations | GPC | GPC with Citations | XGP |
|-------|-----|-----------|-----|--------------------|-----|
| 5vs1 | 346/488 | 2466 | $0.905 \pm 0.031$ | $0.891 \pm 0.022$ | $0.945 \pm 0.053$ |
| 5vs2 | 346/619 | 3417 | $0.900 \pm 0.032$ | $0.905 \pm 0.044$ | $0.933 \pm 0.059$ |
| 5vs3 | 346/1376 | 3905 | $0.863 \pm 0.040$ | $0.893 \pm 0.017$ | $0.883 \pm 0.013$ |
| 5vs4 | 346/646 | 2858 | $0.916 \pm 0.030$ | $0.887 \pm 0.018$ | $0.951 \pm 0.042$ |
| 5vs6 | 346/281 | 1968 | $0.887 \pm 0.054$ | $0.843 \pm 0.076$ | $0.955 \pm 0.041$ |
| 5vs7 | 346/529 | 2948 | $0.869 \pm 0.045$ | $0.867 \pm 0.041$ | $0.926 \pm 0.076$ |

## 4.1 Political books

We consider first a simple classification problem where the goal is to classify whether a particular book is of liberal political inclination or not. The features of each book are given by the words in the Amazon.com front page for that particular book. The choice of books, labels, and relationships are given in the data collected by Valdis Krebs and available at http://www-personal.umich.edu/ mejn/netdata. The data containing book features can be found at http://www.statslab.cam.ac.uk/~silva. There are 105 books, 43 of which are labeled as liberal books. The relationships are pairs of books which are frequently purchased together by a same customer. Notice this is an easy problem, where labels are strongly associated if they share a relationship. We performed evaluation by sampling 100 times from the original pool of books, assigning half of them as trainning data. The evaluation criterion was the area under the curve (AUC) for this binary problem. This is a problem where Method I is suboptimal. Figure 4(a) shows the original binary link matrix. Figure 4(b) depicts the corresponding $\mathbf{U}^0$ matrix obtained with Method I, where entries closer to red correspond to stronger correlations. Method II gives a better performance here (Method I was better in the next two experiments). The AUC result for GPC was of 0.92, while both RGP and XGP achieved 0.98 (the difference between XGP and GPC having a std. deviation of 0.02).

## 4.2 Cora

The Cora collection [6] contains over 50,000 computer science research papers including bibliographic citations. We used a subset in our experiment. The subset consists of 4,285 machine learning papers categorized into 7 classes. The second column of Table 1 shows the class sizes. Each paper was preprocessed as a bag-of-words, a vector of "term frequency" components scaled by "inverse document frequency", and then normalized to unity length. This follows the pre-processing used in [2]. There is a total of 20,082 features. For each class, we randomly selected $1\%$ of the labelled samples for training and tested on the remainder. The partition was repeated 100 times. We used the fact that the database is composed of fairly specialized papers as an illustration of when XGP might not be as optimal as RGP (whose AUC curves are very close to 1), since the population of links tends to be better separated between different classes (but this is also means that the task is fairly easy, and differences disappear very rapidly with increasing sample sizes). The fact there is very little training data also favors RGP, since XGP propagates information through training points. Still, XGP does better than the non-relational GPC. Notice that adding the citation adjacency matrix as a binary input feature for each paper does not improve the performance of the GPC, as shown in Table 1. Results for other classes are of similar qualitative nature and not displayed here.

## 4.3 WebKB

The WebKB dataset consists of homepages from 4 different universities: Cornell, Texas, Washington and Wisconsin [3]. Each webpage belongs to one out of 7 categories: student, professor, course, project, staff, department and "other". The relations come from actual links in the webpages. There is relatively high heterogeneity of types of links in each page: in terms of mixed graph modeling, this linkage mechanism is explained by a hidden common cause (e.g., a student and a course page are associated because that person's interest in enrolling as a student also creates demand for a course). The heterogeneity also suggests that two unlinked pages should not, on average, have an association if they link to a common page $W$. However, observing the type of page $W$ might create

Table 2: Comparison of the three algorithms on the task "other" vs. "not-other" in the WebKB domain. Results for GPC and RGP taken from [2]. The same partitions for training and test are used to generate the results for XGP. Mean and standard deviation of AUC results are reported.

| University | Numbers | | | Other or Not | | |
|---|---|---|---|---|---|---|
| | Other | All | Link | GPC | RGP | XGP |
| Cornell | 617 | 865 | 13177 | $0.708 \pm 0.021$ | $0.884 \pm 0.025$ | $0.917 \pm 0.022$ |
| Texas | 571 | 827 | 16090 | $0.799 \pm 0.021$ | $0.906 \pm 0.026$ | $0.949 \pm 0.015$ |
| Washington | 939 | 1205 | 15388 | $0.782 \pm 0.023$ | $0.877 \pm 0.024$ | $0.923 \pm 0.016$ |
| Wisconsin | 942 | 1263 | 21594 | $0.839 \pm 0.014$ | $0.899 \pm 0.015$ | $0.941 \pm 0.018$ |

the association. We compare how the three algorithms perform when trying to predict if a webpage is of class "other" or not (the other classifications are easier, with smaller differences. Results are omitted for space purposes). The proportion of "other" to non-"other" is about 4:1, which makes the area under the curve (AUC) a more suitable measure of success. We used the same 100 subsamples from [2], where 10% of the whole data is sampled from the pool for a specific university, and the remaining is used for test. We also used the same features as in [2], pre-processed as described in the previous section. The results are shown in Table 2. Both relational Gaussian processes are far better than the non-relational GPC. XGP gives significant improvements over RGP in all four universities.

## 5 Conclusion

We introduced a new family of relational classifiers by extending a classical statistical model [12] to non-parametric relational classification. This is inspired by recent advances in relational Gaussian processes [2] and Bayesian inference for mixed graph models [11]. We showed empirically that modeling the type of latent phenomena that our approach postulates can sometimes improve prediction performance in problems traditionally approached by Markov network structures.

Several interesting problems can be treated in the future. It is clear that there are many different ways by which the relational covariance matrix can be parameterized. Intermediate solutions between Methods I and II, approximations through matrix factorizations and graph cuts are only a few among many alternatives that can be explored. Moreover, there is a relationship between our model and multiple kernel learning [1], where one of the kernels comes from error covariances. This might provide alternative ways of learning our models, including multiple types of relationships.

**Acknowledgements:** We thank Vikas Sindhwani for the preprocessed Cora database.

## Footnotes

[1]For Gaussian models, the absence of an edge in the undirected representation (i.e., Gaussian Markov random fields) corresponds to a zero entry in the *inverse* covariance matrix, where in the DMG it corresponds to a zero in the covariance matrix [9].

[2]In the figure, we are not representing explicitly that $f_1$, $f_2$ and $f_3$ are not independent (the prior covariance matrix $\Sigma$ is complete). The figure is meant as a representation of the extra associations that arise when conditioning on $\xi = 1$, and the way such associations factorize.

[3]The need for an approximation is not a shortcoming only of the DMG approach. Notice that the relational Gaussian process of [2] also requires an approximation of its relational kernel.

[4]We provide MATLAB/Octave code for our method in http://www.statslab.cam.ac.uk/~silva.

[5]For triangulation, we used the MATLAB implementation of the Reverse Cuthill McKee vertex ordering available at http://people.scs.fsu.edu/~burkardt/m_src/rcm/rcm.html

## References

[1] F. Bach, G. Lanckriet, and M. Jordan. Multiple kernel learning, conic duality, and the SMO algorithm. *21st International Conference on Machine Learning*, 2004.

[2] W. Chu, V. Sindhwani, Z. Ghahramani, and S. Keerthi. Relational learning with Gaussian processes. *Neural Information Processing Systems*, 2006.

[3] M. Craven, D. DiPasquo, D. Freitag, A. McCallum, T. Mitchell, K. Nigam, and S. Slattery. Learning to extract symbolic knowledge from the World Wide Web. *Proceedings of AAAI'98*, pages 509–516, 1998.

[4] J. Lafferty, A. McCallum, and F. Pereira. Conditional random fields: Probabilistic models for segmenting and labeling sequence data. *18th International Conference on Machine Learning*, 2001.

[5] S. Lauritzen. *Graphical Models*. Oxford University Press, 1996.

[6] A. McCallum, K. Nigam, J. Rennie, and K. Seymore. Automating the construction of Internet portals with machine learning. *Information Retrieval Journal*, 3:127–163, 2000.

[7] T. Minka. A family of algorithms for approximate Bayesian inference. *PhD Thesis, MIT*, 2001.

[8] C. Rasmussen and C. Williams. *Gaussian Processes for Machine Learning*. MIT Press, 2006.

[9] T. Richardson and P. Spirtes. Ancestral graph Markov models. *Annals of Statistics*, 30:962–1030, 2002.

[10] P. Sen and L. Getoor. Link-based classification. *Report CS-TR-4858, University of Maryland*, 2007.

[11] R. Silva and Z. Ghahramani. Bayesian inference for Gaussian mixed graph models. *UAI*, 2006.

[12] A. Zellner. An efficient method of estimating seemingly unrelated regression equations and tests for aggregation bias. *Journal of the American Statistical Association*, 1962.
